# Eigenvoice Speaker Adaptation via Composite Kernel PCA

**James T. Kwok, Brian Mak and Simon Ho**
Department of Computer Science
Hong Kong University of Science and Technology
Clear Water Bay, Hong Kong
[jamesk,mak,csho]@cs.ust.hk

## Abstract

Eigenvoice speaker adaptation has been shown to be effective when only a small amount of adaptation data is available. At the heart of the method is principal component analysis (PCA) employed to find the most important eigenvoices. In this paper, we postulate that nonlinear PCA, in particular kernel PCA, may be even more effective. One major challenge is to map the feature-space eigenvoices back to the observation space so that the state observation likelihoods can be computed during the estimation of eigenvoice weights and subsequent decoding. Our solution is to compute kernel PCA using composite kernels, and we will call our new method *kernel eigenvoice speaker adaptation*. On the TIDIGITS corpus, we found that compared with a speaker-independent model, our kernel eigenvoice adaptation method can reduce the word error rate by 28–33% while the standard eigenvoice approach can only match the performance of the speaker-independent model.

## 1 Introduction

In recent years, there has been a lot of interest in the study of kernel methods [1]. The basic idea is to map data in the input space $\mathcal{X}$ to a feature space via some nonlinear map $\varphi$, and then apply a linear method there. It is now well known that the computational procedure depends only on the inner products[1] $\varphi(\mathbf{x}_i)'\varphi(\mathbf{x}_j)$ in the feature space (where $\mathbf{x}_i, \mathbf{x}_j \in \mathcal{X}$), which can be obtained efficiently from a suitable kernel function $k(\cdot, \cdot)$. Besides, kernel methods have the important computational advantage that no nonlinear optimization is involved. Thus, the use of kernels provides elegant nonlinear generalizations of many existing linear algorithms. A well-known example in supervised learning is the support vector machines (SVMs). In unsupervised learning, the kernel idea has also led to methods such as kernel-based clustering algorithms and kernel principal component analysis [2].

In the field of automatic speech recognition, eigenvoice speaker adaptation [3] has drawn some attention in recent years as it is found particularly useful when only a small amount of adaptation speech is available; e.g. a few seconds. At the heart of the method is principal component analysis (PCA) employed to find the most important eigenvoices. Then

a new speaker is represented as a linear combination of a few (most important) eigen-voices and the eigenvoice weights are usually estimated by maximizing the likelihood of the adaptation data. Conventionally, these eigenvoices are found by linear PCA. In this paper, we investigate the use of nonlinear PCA to find the eigenvoices by kernel methods. In effect, the nonlinear PCA problem is converted to a linear PCA problem in the high-dimension feature space using the kernel trick. One of the major challenges is to map the feature-space eigenvoices back to the observation space to compute the state observation likelihood of adaptation data during the estimation of eigenvoice weights and likelihood of test data during decoding. Our solution is to compute kernel PCA using composite kernels. We will call our new method *kernel eigenvoice speaker adaptation*.

Kernel eigenvoice adaptation will have to deal with several parameter spaces. To avoid confusion, we denote the several spaces as follows: the $d_1$-dimensional observation space as $\mathcal{O}$; the $d_2$-dimensional speaker (supervector) space as $\mathcal{X}$; and the $d_3$-dimensional speaker feature space as $\mathcal{F}$. Notice that $d_1 \ll d_2 \ll d_3$ in general.

The rest of this paper is organized as follows. Brief overviews on eigenvoice speaker adaptation and kernel PCA are given in Sections 2 and 3. Sections 4 and 5 then describe our proposed kernel eigenvoice method and its robust extension. Experimental results are presented in Section 6, and the last section gives some concluding remarks.

## 2   Eigenvoice

In the standard eigenvoice approach [3], speech training data are collected from many speakers with diverse characteristics. A set of *speaker-dependent* (SD) acoustic hidden Markov models (HMMs) are trained from each speaker where each HMM state is modeled as a mixture of Gaussian distributions. A speaker's voice is then represented by a *speaker supervector* that is composed by concatenating the mean vectors of all HMM Gaussian distributions. For simplicity, we assume that each HMM state consists of one Gaussian only. The extension to mixtures of Gaussians is straightforward. Thus, the $i$th speaker supervector consists of $R$ constituents, one from each Gaussian, and will be denoted by $\mathbf{x}_i = [\mathbf{x}'_{i1} \ldots \mathbf{x}'_{iR}]' \in \mathbb{R}^{d_2}$. The similarity between any two speaker supervectors $\mathbf{x}_i$ and $\mathbf{x}_j$ is measured by their dot product

$$\mathbf{x}'_i \mathbf{x}_j = \sum_{r=1}^{R} \mathbf{x}'_{ir} \mathbf{x}_{jr} \ . \tag{1}$$

PCA is then performed on a set of training speaker supervectors and the resulting eigen-vectors are called *eigenvoices*. To adapt to a new speaker, his/her supervector $\mathbf{s}$ is treated as a linear combination of the first $M$ eigenvoices $\{\mathbf{v}_1, \ldots, \mathbf{v}_M\}$, i.e., $\mathbf{s} = \mathbf{s}^{(ev)} = \sum_{m=1}^{M} w_m \mathbf{v}_m$ where $\mathbf{w} = [w_1, \ldots, w_M]'$ is the eigenvoice weight vector. Usually, only a few eigenvoices (e.g., $M < 10$) are employed so that a little amount of adaptation speech (e.g., a few seconds) will be required. Given the adaptation data $\mathbf{o}_t, t = 1, \ldots, T$, the eigenvoice weights are in turn estimated by maximizing the likelihood of the $\mathbf{o}_t$'s. Mathematically, one finds $\mathbf{w}$ by *maximizing* the $Q$ function: $Q(\mathbf{w}) = Q_\pi + Q_a + Q_b(\mathbf{w})$, where

$$Q_\pi = \sum_{r=1}^{R} \gamma_1(r) \log(\pi_r) , \quad Q_a = \sum_{p,r=1}^{R} \sum_{t=1}^{T-1} \xi_t(p,r) \log(a_{pr}) ,$$

$$\text{and,} \quad Q_b(\mathbf{w}) = \sum_{r=1}^{R} \sum_{t=1}^{T} \gamma_t(r) \log(b_r(\mathbf{o}_t, \mathbf{w})) , \tag{2}$$

and $\pi_r$ is the initial probability of state $r$; $\gamma_t(r)$ is the posterior probability of observation sequence being at state $r$ at time $t$; $\xi_t(p,r)$ is the posterior probability of observation sequence being at state $p$ at time $t$ and at state $r$ at time $t+1$; $b_r$ is the Gaussian pdf of the $r$th

state after re-estimation. Furthermore, $Q_b$ is related to the new speaker supervector $\mathbf{s}$ by

$$Q_b(\mathbf{w}) = -\frac{1}{2} \sum_{r=1}^{R} \sum_{t=1}^{T} \gamma_t(r) \left[ d_1 \log(2\pi) + \log |\mathbf{C}_r| + \|\mathbf{o}_t - \mathbf{s}_r(\mathbf{w})\|_{\mathbf{C}_r}^2 \right] , \qquad (3)$$

where $\|\mathbf{o}_t - \mathbf{s}_r(\mathbf{w})\|_{\mathbf{C}_r}^2 = (\mathbf{o}_t - \mathbf{s}_r(\mathbf{w}))' \mathbf{C}_r^{-1} (\mathbf{o}_t - \mathbf{s}_r(\mathbf{w}))$ and $\mathbf{C}_r$ is the covariance matrix of the Gaussian at state $r$.

## 3 Kernel PCA

In this paper, the computation of eigenvoices is generalized by performing kernel PCA instead of linear PCA. In the following, let $k(\cdot, \cdot)$ be the kernel with associated mapping $\varphi$ which maps a pattern $\mathbf{x}$ in the speaker supervector space $\mathcal{X}$ to $\varphi(\mathbf{x})$ in the speaker feature space $\mathcal{F}$. Given a set of $N$ patterns (speaker supervectors) $\{\mathbf{x}_1, \ldots, \mathbf{x}_N\}$, denote the mean of the $\varphi$-mapped feature vectors by $\bar{\varphi} = \frac{1}{N} \sum_{i=1}^{N} \varphi(\mathbf{x}_i)$, and the "centered" map by $\tilde{\varphi}$ (with $\tilde{\varphi}(\mathbf{x}) = \varphi(\mathbf{x}) - \bar{\varphi}$). Eigendecomposition is performed on $\tilde{\mathbf{K}}$, the centered version of $\mathbf{K} = [k(\mathbf{x}_i, \mathbf{x}_j)]_{ij}$, as $\tilde{\mathbf{K}} = \mathbf{U}\mathbf{\Lambda}\mathbf{U}'$, where $\mathbf{U} = [\boldsymbol{\alpha}_1, \ldots, \boldsymbol{\alpha}_N]$ with $\boldsymbol{\alpha}_i = [\alpha_{i1}, \ldots, \alpha_{iN}]'$, and $\mathbf{\Lambda} = \mathrm{diag}(\lambda_1, \ldots, \lambda_N)$. Notice that $\tilde{\mathbf{K}}$ is related to $\mathbf{K}$ by $\tilde{\mathbf{K}} = \mathbf{H}\mathbf{K}\mathbf{H}$, where $\mathbf{H} = \mathbf{I} - \frac{1}{N}\mathbf{1}\mathbf{1}'$ is the centering matrix, $\mathbf{I}$ is the $N \times N$ identity matrix, and $\mathbf{1} = [1, \ldots, 1]'$, an $N$-dimensional vector. The $m$th orthonormal eigenvector of the covariance matrix in the feature space is then given by [2] as $\mathbf{v}_m = \sum_{i=1}^{N} \frac{\alpha_{mi}}{\sqrt{\lambda_m}} \tilde{\varphi}(\mathbf{x}_i)$ .

## 4 Kernel Eigenvoice

As seen from Eqn (3), the estimation of eigenvoice weights requires the evaluation of the distance between adaptation data $\mathbf{o}_t$ and Gaussian means of the new speaker in the observation space $\mathcal{O}$. In the standard eigenvoice method, this is done by first breaking down the adapted speaker supervector $\mathbf{s}$ to its $R$ constituent Gaussians $\mathbf{s}_1, \ldots, \mathbf{s}_R$. However, the use of kernel PCA does not allow us to access each constituent Gaussians directly. To get around the problem, we investigate the use of composite kernels.

### 4.1 Definition of the Composite Kernel

For the $i$th speaker supervector $\mathbf{x}_i$, we map each constituent $\mathbf{x}_{ir}$ separately via a kernel $k_r(\cdot, \cdot)$ to $\varphi_r(\mathbf{x}_{ir})$, and then construct $\varphi(\mathbf{x}_i)$ as $\varphi(\mathbf{x}_i) = [\varphi_1(\mathbf{x}_{i1})', \ldots, \varphi_R(\mathbf{x}_{iR})']'$. Analogous to Eqn (1), the similarity between two speaker supervectors $\mathbf{x}_i$ and $\mathbf{x}_j$ in the composite feature space is measured by

$$k(\mathbf{x}_i, \mathbf{x}_j) = \sum_{r=1}^{R} k_r(\mathbf{x}_{ir}, \mathbf{x}_{jr}) .$$

Note that if $k_r$'s are valid Mercer kernels, so is $k$ [1].

Using this composite kernel, we can then proceed with the usual kernel PCA on the set of $N$ training speaker supervectors and obtain $\boldsymbol{\alpha}_m$'s, $\lambda_m$'s, and the orthonormal eigenvectors $\mathbf{v}_m$'s ($m = 1, \ldots, M$) of the covariance matrix in the feature space $\mathcal{F}$.

### 4.2 New Speaker in the Feature Space

In the following, we denote the supervector of a new speaker by $\mathbf{s}$. Similar to the standard eigenvoice approach, its $\tilde{\varphi}$-mapped speaker feature vector[2] $\tilde{\varphi}^{(kev)}(\mathbf{s})$ is assumed to be a

linear combination of the first $M$ eigenvectors, i.e.,

$$\tilde{\varphi}^{(kev)}(\mathbf{s}) = \sum_{m=1}^{M} w_m \mathbf{v}_m = \sum_{m=1}^{M} \sum_{i=1}^{N} \frac{w_m \alpha_{mi}}{\sqrt{\lambda_m}} \tilde{\varphi}(\mathbf{x}_i). \tag{4}$$

Its $r$th constituent is then given by

$$\tilde{\varphi}_r^{(kev)}(\mathbf{s}_r) = \sum_{m=1}^{M} \sum_{i=1}^{N} \frac{w_m \alpha_{mi}}{\sqrt{\lambda_m}} \tilde{\varphi}_r(\mathbf{x}_{ir}) \ .$$

Hence, the similarity between $\varphi_r^{(kev)}(\mathbf{s}_r)$ and $\varphi_r(\mathbf{o}_t)$ is given by

$$
\begin{aligned}
k_r^{(kev)}(\mathbf{s}_r, \mathbf{o}_t) &\equiv \varphi(\mathbf{s}_r)' \varphi_r(\mathbf{o}_t) \\
&= \left[ \left( \sum_{m=1}^{M} \sum_{i=1}^{N} \frac{w_m \alpha_{mi}}{\sqrt{\lambda_m}} \tilde{\varphi}_r(\mathbf{x}_{ir}) \right) + \bar{\varphi}_r \right]' \varphi_r(\mathbf{o}_t) \\
&= \left[ \left( \sum_{m=1}^{M} \sum_{i=1}^{N} \frac{w_m \alpha_{mi}}{\sqrt{\lambda_m}} (\varphi_r(\mathbf{x}_{ir}) - \bar{\varphi}_r) \right) + \bar{\varphi}_r \right]' \varphi_r(\mathbf{o}_t) \\
&= \sum_{m=1}^{M} \sum_{i=1}^{N} \frac{w_m \alpha_{mi}}{\sqrt{\lambda_m}} (k_r(\mathbf{x}_{ir}, \mathbf{o}_t) - \bar{\varphi}_r' \varphi_r(\mathbf{o}_t)) + \bar{\varphi}_r' \varphi_r(\mathbf{o}_t) \\
&\equiv A(r,t) + \sum_{m=1}^{M} \frac{w_m}{\sqrt{\lambda_m}} B(m,r,t), \tag{5}
\end{aligned}
$$

where $\bar{\varphi}_r = \frac{1}{N} \sum_{i=1}^{N} \varphi_r(\mathbf{x}_{ir})$ is the $r$th part of $\bar{\varphi}$,

$$A(r,t) = \bar{\varphi}_r' \varphi_r(\mathbf{o}_t) = \frac{1}{N} \sum_{j=1}^{N} k_r(\mathbf{x}_{jr}, \mathbf{o}_t),$$

and

$$B(m,r,t) = \left( \sum_{i=1}^{N} \alpha_{mi} k_r(\mathbf{x}_{ir}, \mathbf{o}_t) \right) - A(r,t) \left( \sum_{i=1}^{N} \alpha_{mi} \right).$$

### 4.3  Maximum Likelihood Adaptation Using an Isotropic Kernel

On adaptation, we have to express $\|\mathbf{o}_t - \mathbf{s}_r\|_{\mathbf{C}_r}^2$ of Eqn (3) as a function of $\mathbf{w}$. Consider using isotropic kernels for $k_r$ so that $k_r(\mathbf{x}_{ir}, \mathbf{x}_{jr}) = \kappa(\|\mathbf{x}_{ir} - \mathbf{x}_{jr}\|_{\mathbf{C}_r})$. Then $k_r^{(kev)}(\mathbf{s}_r, \mathbf{o}_t) = \kappa(\|\mathbf{o}_t - \mathbf{s}_r\|_{\mathbf{C}_r}^2)$, and if $\kappa$ is invertible, $\|\mathbf{o}_t - \mathbf{s}_r\|_{\mathbf{C}_r}^2$ will be a function of $k_r^{(kev)}(\mathbf{s}_r, \mathbf{o}_t)$, which in turn is a function of $\mathbf{w}$ by Eqn (5). In the sequel, we will use the Gaussian kernel $k_r(\mathbf{x}_{ir}, \mathbf{x}_{jr}) = \exp(-\beta_r \|\mathbf{x}_{ir} - \mathbf{x}_{jr}\|_{\mathbf{C}_r}^2)$, and hence

$$\|\mathbf{o}_t - \mathbf{s}_r\|_{\mathbf{C}_r}^2 = -\frac{1}{\beta_r} \log k_r^{(kev)}(\mathbf{s}_r, \mathbf{o}_t) = -\frac{1}{\beta_r} \log \left( A(r,t) + \sum_{m=1}^{M} \frac{w_m}{\sqrt{\lambda_m}} B(m,r,t) \right). \tag{6}$$

Substituting Eqn (6) for the $Q_b$ function in Eqn (3), and differentiating with respect to each eigenvoice weight, $w_j, j = 1, \ldots, M$, we obtain

$$\frac{\partial Q_b}{\partial w_j} = \frac{1}{2\sqrt{\lambda_j}} \sum_{r=1}^{R} \sum_{t=1}^{T} \frac{\gamma_t(r)}{\beta_r} \cdot \frac{B(j,r,t)}{k_r^{(kev)}(\mathbf{s}_r, \mathbf{o}_t)}. \tag{7}$$

not exist, its notation as $\tilde{\varphi}^{(kev)}(\mathbf{s})$ is not exactly correct. However, the notation is adopted for its intuitiveness and the readers are advised to infer the existence of $\mathbf{s}$ based on the context.

Since $Q_\pi$ and $Q_a$ do not depend on $\mathbf{w}$, $\dfrac{\partial Q}{\partial w_j} = \dfrac{\partial Q_b}{\partial w_j}$.

### 4.4 Generalized EM Algorithm

Because of the nonlinear nature of kernel PCA, Eqn (6) is nonlinear in $\mathbf{w}$ and there is no closed form solution for the optimal $\mathbf{w}$. In this paper, we instead apply the generalized EM algorithm (GEM) [4] to find the optimal weights. GEM is similar to standard EM except for the maximization step: EM looks for $\mathbf{w}$ that maximizes the expected likelihood of the E-step but GEM only requires a $\mathbf{w}$ that improves the likelihood. Many numerical methods may be used to update $\mathbf{w}$ based on the derivatives of $Q$. In this paper, gradient ascent is used to get $\mathbf{w}(n)$ from $\mathbf{w}(n-1)$ based only on the first-order derivative as: $\mathbf{w}(n) = \mathbf{w}(n-1) + \eta(n)\mathbf{Q}'|_{\mathbf{w}=\mathbf{w}(n-1)}$, where $\mathbf{Q}' = \dfrac{\partial Q_b}{\partial \mathbf{w}}$ and $\eta(n)$ is the learning rate at the $n$th iteration. Methods such as the Newton's method that uses the second-order derivatives may also be used for faster convergence, at the expense of computing the more costly Hessian in each iteration.

The initial value of $\mathbf{w}(0)$ can be important for numerical methods like gradient ascent. One reasonable approach is to start with the eigenvoice weights of the supervector composed from the speaker-independent model $\mathbf{x}^{(si)}$. That is,

$$
\begin{aligned}
\mathbf{w}_m &= \mathbf{v}'_m \tilde{\varphi}(\mathbf{x}^{(si)}) = \sum_{i=1}^{N} \frac{\alpha_{mi}}{\sqrt{\lambda_m}} \tilde{\varphi}(\mathbf{x}_i)' \tilde{\varphi}(\mathbf{x}^{(si)}) = \sum_{i=1}^{N} \frac{\alpha_{mi}}{\sqrt{\lambda_m}} [\varphi(\mathbf{x}_i) - \bar{\varphi}]' [\varphi(\mathbf{x}^{(si)}) - \bar{\varphi}] \\
&= \sum_{i=1}^{N} \frac{\alpha_{mi}}{\sqrt{\lambda_m}} \left[ k(\mathbf{x}_i, \mathbf{x}^{(si)}) + \frac{1}{N^2} \sum_{p,q=1}^{N} k(\mathbf{x}_p, \mathbf{x}_q) - \frac{1}{N} \sum_{p=1}^{N} \big( k(\mathbf{x}_i, \mathbf{x}_p) + k(\mathbf{x}^{(si)}, \mathbf{x}_p) \big) \right] .
\end{aligned}
\tag{8}
$$

## 5 Robust Kernel Eigenvoice

The success of the eigenvoice approach for fast speaker adaptation is due to two factors: (1) a good collection of "diverse" speakers so that the whole speaker space is captured by the eigenvoices; and (2) the number of adaptation parameters is reduced to a few eigenvoice weights. However, since the amount of adaptation data is so little the adaptation performance may vary widely. To get a more robust performance, we propose to interpolate the kernel eigenvoice $\tilde{\varphi}^{(kev)}(\mathbf{s})$ obtained in Eqn (4) with the $\tilde{\varphi}$-mapped speaker-independent (SI) supervector $\tilde{\varphi}(\mathbf{x}^{(si)})$ to obtain the final speaker adapted model $\tilde{\varphi}^{(rkev)}(\mathbf{s})$ as follows:

$$
\tilde{\varphi}^{(rkev)}(\mathbf{s}) = w_0 \tilde{\varphi}(\mathbf{x}^{(si)}) + (1 - w_0) \tilde{\varphi}^{(kev)}(\mathbf{s}), \quad 0.0 \le w_0 \le 1.0 ,
\tag{9}
$$

where $\tilde{\varphi}^{(kev)}(\mathbf{s})$ is found by Eqn (4). By replacing $\tilde{\varphi}^{(kev)}(\mathbf{s})$ by $\tilde{\varphi}^{(rkev)}(\mathbf{s})$ for the computation of the kernel value of Eqn (5), and following the mathematical steps in Section 4, one may derive the required gradients for the joint maximum-likelihood estimation of $w_0$ and other eigenvoice weights in the GEM algorithm.

Notice that $\tilde{\varphi}^{(rkev)}(\mathbf{s})$ also contains components in $\tilde{\varphi}(\mathbf{x}^{(si)})$ from eigenvectors beyond the $M$ selected kernel eigenvoices for adaptation. Thus, robust KEV adaptation may have the additional benefit of preserving the speaker-independent projections on the remaining less important but robust eigenvoices in the final speaker-adapted model.

# 6 Experimental Evaluation

The proposed kernel eigenvoice adaptation method was evaluated on the TIDIGITS speech corpus [5]. Its performance was compared with that of the speaker-independent model and the standard eigenvoice adaptation method using only 3s, 5.5s, and 13s of adaptation speech. If we exclude the leading and ending silence, the average duration of adaptation speech is 2.1s, 4.1s, and 9.6s respectively.

## 6.1 TIDIGITS Corpus

The TIDIGITS corpus contains clean connected-digit utterances sampled at 20 kHz. It is divided into a standard training set and a test set. There are 163 speakers (of both genders) in each set, each pronouncing 77 utterances of one to seven digits (out of the eleven digits: "0", "1", …, "9", and "oh".). The speaker characteristics is quite diverse with speakers coming from 22 dialect regions of USA and their ages ranging from 6 to 70 years old.

In all the following experiments, only the training set was used to train the speaker-independent (SI) HMMs and speaker-dependent (SD) HMMs from which the SI and SD speaker supervectors were derived.

## 6.2 Acoustic Models

All training data were processed to extract 12 mel-frequency cepstral coefficients and the normalized frame energy from each speech frame of 25 ms at every 10 ms. Each of the eleven digit models was a strictly left-to-right HMM comprising 16 states and one Gaussian with diagonal covariance per state. In addition, there were a 3-state "sil" model to capture silence speech and a 1-state "sp" model to capture short pauses between digits. All HMMs were trained by the EM algorithm. Thus, the dimension of the observation space $d_1$ is 13 and that of the speaker supervector space $d_2 = 11 \times 16 \times 13 = 2288$.

Firstly, the SI models were trained. Then an SD model was trained for each individual speaker by borrowing the variances and transition matrices from the corresponding SI models, and only the Gaussian means were estimated. Furthermore, the sil and sp models were simply copied to the SD model.

## 6.3 Experiments

The following five models/systems were compared:

**SI:** speaker-independent model

**EV:** speaker-adapted model found by the standard eigenvoice adaptation method.

**Robust-EV:** speaker-adapted models found by our robust version of EV, which is the interpolation between the SI supervector and the supervector found by EV. That is,

$$\mathbf{s}^{(rev)} = w_0 \mathbf{s}^{(si)} + (1 - w_0)\mathbf{s}^{(ev)} , \quad 0.0 \leq w_0 \leq 1.0 .$$

**KEV:** speaker-adapted model found by our new kernel eigenvoice adaptation method as described in Section 4.

**Robust-KEV:** speaker-adapted model found by our robust KEV as described in Section 5.

All adaptation results are the averages of 5-fold cross-validation taken over all 163 test speaker data. The detailed results using different numbers of eigenvoices are shown in Figure 1, while the best result for each model is shown in Table 1.

Table 1: Word recognition accuracies of SI model and the best adapted models found by
EV, robust EV, KEV, and robust KEV using 2.1s, 4.1s, and 9.6s of adaptation speech.

| SYSTEM | 2.1s | 4.1s | 9.6s |
|---|---|---|---|
| SI | 96.25 | | |
| EV | 95.61 | 95.65 | 95.67 |
| robust EV | 96.26 | 96.26 | 96.27 |
| KEV | 96.85 | 97.05 | 97.05 |
| robust KEV | 97.28 | 97.44 | 97.50 |

From Table 1, we observe that the standard eigenvoice approach cannot obtain better perfor-
mance than the SI model[3]. On the other hand, using our kernel eigenvoice (KEV) method,
we obtain a word error rate (WER) reduction of 16.0%, 21.3%, and 21.3% with 2.1s, 4.1s,
and 9.6s of adaptation speech over the SI model. When the SI model is interpolated with
the KEV model in our robust KEV method, the WER reduction further improves to 27.5%,
31.7%, and 33.3% respectively. These best results are obtained with 7 to 8 eigenvoices. The
results show that nonlinear PCA using composite kernels can be more effective in finding
the eigenvoices.

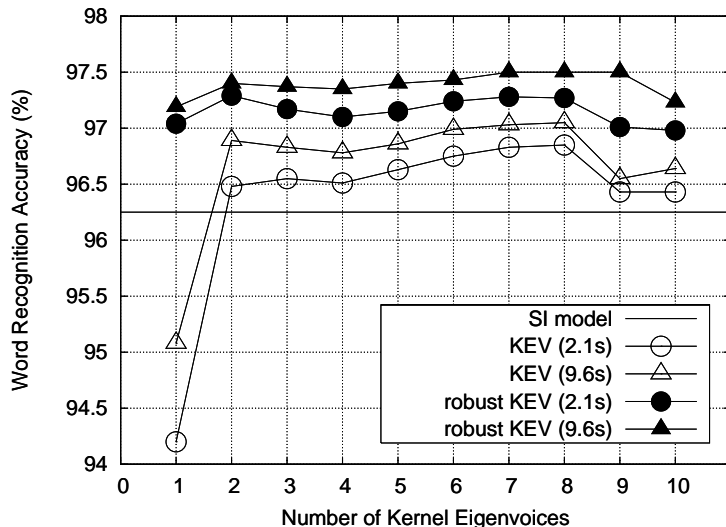

Figure 1: Word recognition accuracies of adapted models found by KEV and robust KEV
using different numbers of eigenvoices.

From Figure 1, the KEV method can outperform the SI model even with only two eigen-
voices using only 2.1s of speech. Its performance then improves slightly with more eigen-
voices or more adaptation data. If we allow interpolation with the SI model as in robust

is about 99.7%. The main reasons are that we used only 13-dimensional static cepstra and energy, and
each state was modelled by a single Gaussian with diagonal covariance. The use of this simple model
allowed us to run experiments with 5-fold cross-validation using very short adaptation speech. Right
now our approach requires computation of many kernel function values and is very computationally
expensive. As a first attempt on the approach, we feel that the use of this simple model is justified.
We are now working on its speed-up and its extension to HMM states of Gaussian mixtures.

KEV, the saturation effect is even more pronounced: even with one eigenvoice, the adaptation performance is already better than that of SI model, and then the performance does not change much with more eigenvoices or adaptation data. The results seem to suggest that the requirement that the adapted speaker supervector is a weighted sum of few eigenvoices is both the strength and weakness of the method: on the one hand, fast adaptation becomes possible since the number of estimation parameters is small, but adaptation saturates quickly because the constraint is so restrictive that all mean vectors of different acoustic models have to undergo the same linear combination of the eigenvoices.

# 7   Conclusions

In this paper, we improve the standard eigenvoice speaker adaptation method using kernel PCA with a composite kernel. In the TIDIGITS task, it is found that while the standard eigenvoice approach does not help, our kernel eigenvoice method may outperform the speaker-independent model by about 28–33% (in terms of error rate improvement).

Right now the speed of recognition using the adapted model that resulted from our kernel eigenvoice method is slower than that from the standard eigenvoice method because any state observation likelihoods cannot be directly computed but through evaluating the kernel values with all training speaker supervectors. One possible solution is to apply sparse kernel PCA [6] so that computation of the first $M$ principal components involves only $M$ (instead of $N$ with $M \ll N$) kernel functions. Another direction is to use compactly supported kernels [7], in which the value of $\kappa(\|\mathbf{x}_i - \mathbf{x}_j\|)$ vanishes when $\|\mathbf{x}_i - \mathbf{x}_j\|$ is greater than a certain threshold. The kernel matrix then becomes sparse. Moreover, no more computation is required when $\|\mathbf{x}_i - \mathbf{x}_j\|$ is large.

# 8   Acknowledgements

This research is partially supported by the Research Grants Council of the Hong Kong SAR under the grant numbers HKUST2033/00E, HKUST6195/02E, and HKUST6201/02E.

## Footnotes

[1]In this paper, vector/matrix transpose is denoted by the superscript $'$.

[2]The notation for a new speaker in the feature space requires some explanation. If $\mathbf{s}$ exists, then its centered image is $\tilde{\varphi}^{(kev)}(\mathbf{s})$. However, since the pre-image of a speaker in the feature space may

[3]The word accuracy of our SI model is not as good as the best reported result on TIDIGITS which

# References

[1] B. Schölkopf and A.J. Smola. *Learning with Kernels*. MIT, 2002.

[2] B. Schölkopf, A. Smola, and K.R. Müller. Nonlinear component analysis as a kernel eigenvalue problem. *Neural Computation*, 10:1299–1319, 1998.

[3] R. Kuhn, J.-C. Junqua, P. Nguyen, and N. Niedzielski. Rapid Speaker Adaptation in Eigenvoice Space. *IEEE Transactions on Speech and Audio Processing*, 8(4):695–707, Nov 2000.

[4] A.P. Dempster, N.M. Laird, and D.B. Rubin. Maximum likelihood from incomplete data via the EM algorithm. *Journal of the Royal Statistical Society: Series B*, 39(1):1–38, 1977.

[5] R.G. Leonard. A Database for Speaker-Independent Digit Recognition. In *Proceedings of the IEEE International Conference on Acoustics, Speech, and Signal Processing*, volume 3, pages 4211–4214, 1984.

[6] A.J. Smola, O.L. Mangasarian, and B. Schölkopf. Sparse kernel feature analysis. Technical Report 99-03, Data Mining Institute, University of Wisconsin, Madison, 1999.

[7] M.G. Genton. Classes of kernels for machine learning: A statistics perspective. *Journal of Machine Learning Research*, 2:299–312, 2001.
